# Adaptive Range Coding

Bruce E. Rosen, James M. Goodwin, and Jacques J. Vidal
Distributed Machine Intelligence Laboratory
Computer Science Department
University of California, Los Angeles
Los Angeles, CA 90024

## Abstract

This paper examines a class of neuron based learning systems for dynamic control that rely on adaptive range coding of sensor inputs. Sensors are assumed to provide binary coded range vectors that coarsely describe the system state. These vectors are input to neuron-like processing elements. Output decisions generated by these "neurons" in turn affect the system state, subsequently producing new inputs. Reinforcement signals from the environment are received at various intervals and evaluated. The neural weights as well as the *range boundaries* determining the output decisions are then altered with the goal of maximizing future reinforcement from the environment. Preliminary experiments show the promise of adapting "neural receptive fields" when learning dynamical control. The observed performance with this method exceeds that of earlier approaches.

# 1  INTRODUCTION

A major criticism of unsupervised learning and control techniques such as those used by Barto et al. (Barto, 1983) and by Albus (Albus, 1981) is the need for *a priori* selection of region sizes for range coding. Range coding in principle generalizes inputs and reduces computational and storage overhead, but the boundary partitioning, determined *a priori*, is often non-optimal (for example, the ranges described in (Barto, 1983) differ from those used in (Barto 1982) for the same control task differ). Determination of nearly optimal, or at least adequate, regions is left as an additional task that would require that the system dynamics be analyzed, which is not always possible.

To address this problem, we move region boundaries adaptively, progressively altering the initial partitioning to a more appropriate representation with no need for *a priori* knowledge. Unlike previous work (Michie, 1968), (Barto, 1983), (Anderson, 1982) which used fixed coders, this approach produces adaptive coders that contract and expand regions/ranges. During adaptation, frequently active regions/ranges contract, reducing the number of situations in which they will be activated, and increasing the chances that neighboring regions will receive input instead. This class of self-organization is discussed in Kohonen (Kohonen, 1984), (Ritter, 1986, 1988). The resulting self-organizing mapping will tend to track the environmental input probability density function. Adaptive range coding creates a focusing mechanism. Resources are distributed according to regional activity level. More resources can be allocated to critical areas of the state space. Concentrated activity is more finely discriminated and corresponding control decisions are more finely tuned.

Dynamic shaping of the region boundaries can be achieved without sacrificing memory or learning speed. Also, since the region boundaries are finally determined solely by the environmental dynamics, optimal *a priori* ranges and region specifications are not necessary.

As an example, consider a one dimensional state space, as shown in figures 1a and 1b. It is is partitioned into three regions by the vertical lines shown. The heavy curve indicates a theoretical optimal control surface (unknown *a priori*) of a state space which the weight in each region should approximate. The dashed horizontal lines show the best learned weight values for the

respective partitionings.    Weight values approximate the mean value of the true control surface weight in each of the regions.

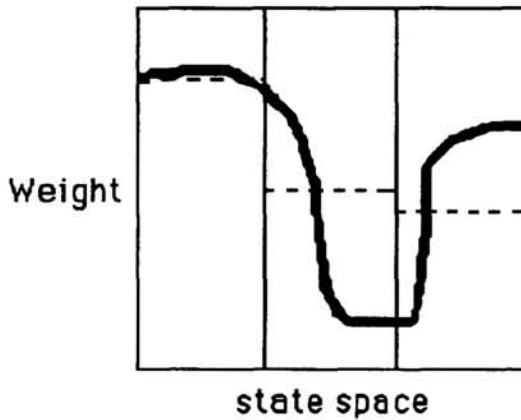

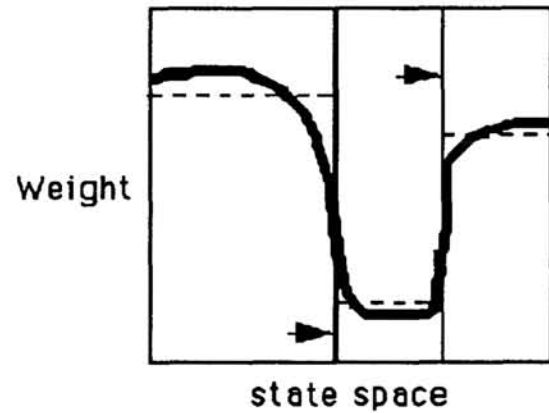

Figure 1a
Even Region Partition

Figure 1b
Adapted Region Partition

An evenly partitioned space produces the weights shown in figure 1a.    Figure 1b shows the regions after the boundaries have been adjusted, and the final weight values.    Although the weights in both 1a and 1b reflect the mean of the true control surface (in their respective regions), adaptive partitioning is able to represent the ideal surface with a smaller mean squared error.

## 2  ADAPTIVE RANGE CODING RULE

For the more general n dimensional control problem using adaptive range boundaries, the shape of each region can change from an initial n dimensional prism to an n dimensional polytope.    The polytope shape is determined by the current activation state and its average activity.    The heuristic for our adaptive range coding is to *move each region vertex towards or away from the current activation state according to the reinforcement.*    The equation which adjusts each region boundary is adapted in part from the weight alteration formula used by Kohonen's topological mapping (Kohonen 1984).    Each region (i) consists of $2^n$ vertices ($V_{ij}(t)$, $1 \leq j \leq 2^n$) describing that region's boundaries that move toward or away from the current state activity (A(t)) depending on the reinforcement r.

[1]        $V_{ij}(t+1) = V_{ij}(t) + K r h(V_{ij}(t) - A(t))$

where K is the gain, r is the reinforcement (or error) used to alter the weight in the region, and h() is a Gaussian or a difference of Gaussians function.

# 3   SIMULATION RESULTS

In our experiments, the expected reinforcement of the ASE/ACE system $\hat{r}$ (described in (Barto 1983)) was also used as r in [1]. Simple pole balancing (see figure 2) was chosen, rather than the cart-pole balancing task in (Barto 1983). The time step $\tau$ was chosen to be large (0.05 seconds) and initial region boundaries of $\theta$ and $\dot{\theta}$ were chosen as (-12,-6,0,1,6,12) and (-∞, -10,10,∞). All other parameters were identical to those described in (Barto, 1983).

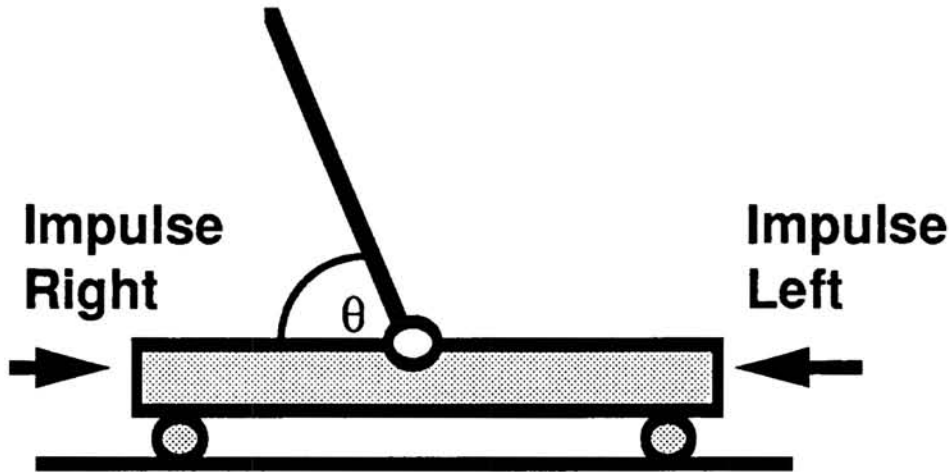

Figure 2:   The Pole Balancing Task

The standard ASE, ASE/ACE, and adaptive range coding algorithms were compared on this task.   One hundred runs of each algorithm were performed.   Each *run* consisted of a sequence of trials and each *trial* counted the number of time steps until the pole fell.   If the pole had not fallen after 20,000 time steps, the trial was considered to be *successful* and it was terminated.   Each run was terminated either after 100 trials, or after the pole was successfully balanced in five successive trials.   (We assumed that five successive trials indicated that the systems weights and regions had stabilized.)   All region weights were initialized to zero at the start of each run.

In the adaptive range coding runs, the updated vertex state positions were determined by 3 factors: difference between the vertex and the current state, the expected reinforcement, and the gain.   A Gaussian served as an appropriate decay function to modulate vertex movements.   Current state to vertex differences served as function input parameters.   Outputs attenuated with

increasing inputs, and the standard deviation σ of the Gaussian shaped the decay function. The magnitude and position of each vertex movement were also modulated by the reinforcement $\hat{r}(t)$ which moves the vertex towards or away form the current state, and by K, a gain parameter. The user definable parameter values of K and σ were initially chosen (arbitrarily) as K = 1 and σ = 10.0, and were used in the following experiments. Parameters were not fine tuned or optimized.

Figure 3 shows the results of the ASE, ASE/ACE, and adaptive range coding experiments. The various runs and trials differed only in the random number generator seed. Corresponding runs and trials using the standard ASE, ASE/ACE and the adaptive range coding algorithm used the same random number seed. All other parameters were identical between the two systems. However, in adaptive range coding, region boundaries were shifted in accordance with [1] during each run.

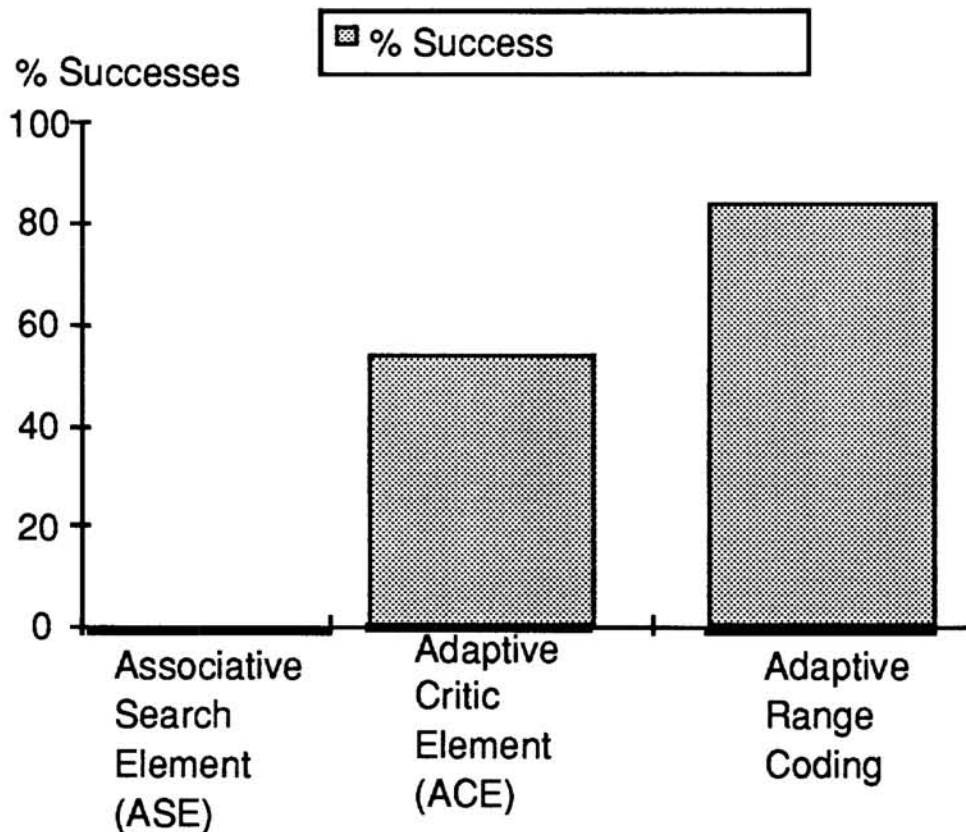

Figure 3:  Comparison of the ASE, ASE/ACE, and the Adaptive Range Coding Algorithm.

We simulated 100 runs of the ASE algorithm with zero successful runs. Using the ASE/ACE algorithm, 54 runs were successful. With adaptive range coding algorithm, 84 of the 100 runs were successful. With $\sigma_{ase/ace} = 4.98$ and $\sigma_{adapt\_range\_code} = 3.66$, a $\chi^2$ test showed the two performance sets to be statistically different ($p > 0.95$).

Figure 4 shows a comparison of the average performance values of the 100 ASE/ACE and Adaptive Range Coding (ARC) runs. Pole balancing time is shown as a function of the number of learning trials experienced.

Pole Balancing Average Performances

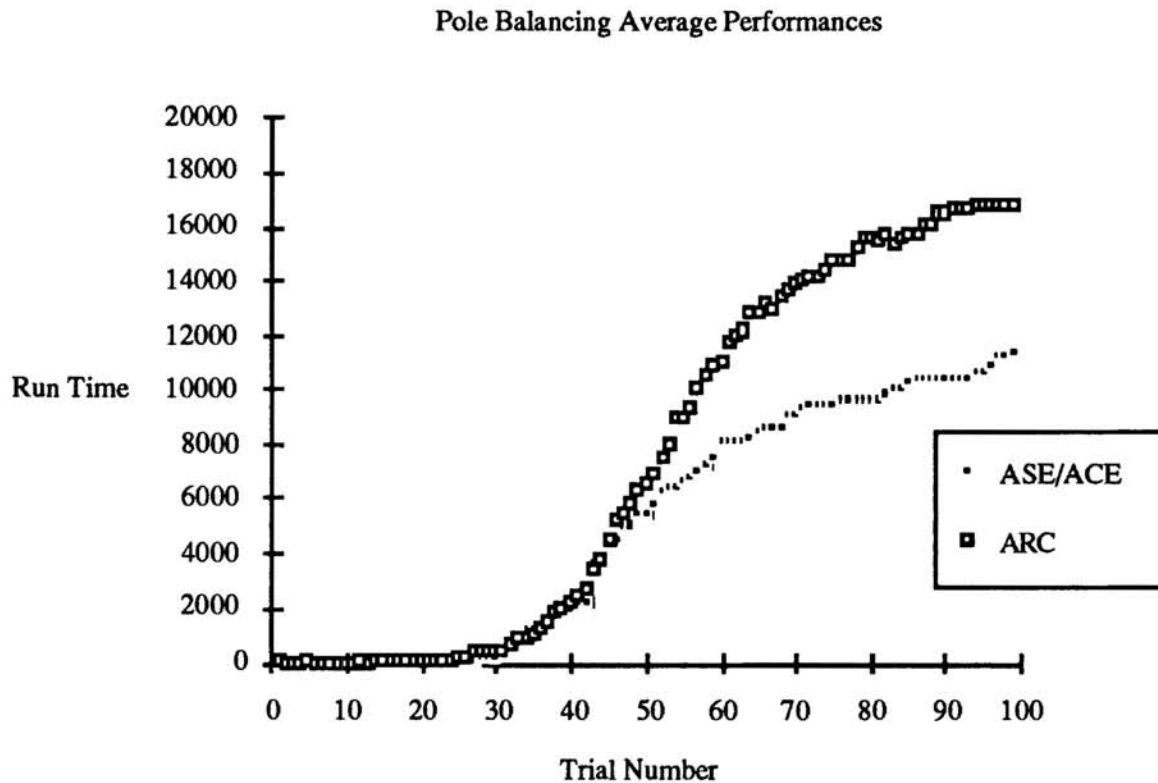

Figure 4: Comparison of the ASE/ACE and Adaptive Range Coding learning rates on the cart pole task. Pole balancing time is shown as function of learning trials. Results are averaged over 100 runs.

The disparity between the run times of the two different algorithms is due to the comparatively large number of failures of the ASE/ACE system. Statistical analysis indicates no significant difference in the learning rates or performance levels of the successful runs between categories, leading us to believe that adaptive range coding may lead to an "all or none"

behavior, and that there is a minimum area of the state space that the system must explore to succeed.

# 4  CONCLUSION

The research has shown that neuron-like elements with adjustable regions can dynamically create topological cause and effect maps reflecting the control laws of dynamic systems. It is anticipated from the results of the examples presented above, that adaptive range coding will be more effective than earlier static region approaches in the control of complex systems with unknown dynamics.

## References

J. S. Albus. (1981) *Brains, Behavior, and Robotics*, Peterburough, NH: McGraw-Hill Byte Books.

C. W. Anderson. (1982) Feature generation and Selection by a Layered Network of Reinforcement Learning Elements: Some Initial Experiments, *Technical Report COINS 82-12*. Amherst, MA: University of Massachusetts, Department of Computer and Information Science.

A. Barto, R. Sutton, and C. Anderson. (1982) Neuron-like elements that can solve difficult learning control problems. *Coins Tech. Rept. No. 82-20*. Amherst, MA: University of Massachusetts, Department of Computer and Information Science.

A. G. Barto, R. S. Sutton, and C. W. Anderson. (1983) Neuron-like elements that can solve difficult learning control problems, *IEEE Transactions on Systems, Man, and Cybernetics*, 13(5): 834-846.

T. Kohonen. (1984) *Self-Organization and Associative Memory*, New York: Springer-Verlag.

D. Michie and R. Chambers. (1968) *Machine Intelligence* Edinburgh: Oliver and Boyd.

H. Ritter and K. Schulten. (1986) Topology Conserving Mappings for Learning Motor Tasks. In J. S. Denker (ed.), *Neural Networks for Computing*. Snowbird, Utah: AIP.

H. Ritter and K. Schulten. (1988) Extending Kohonen's Self-Organizing Mapping Algorithm to Learn Ballistic Movements. In R. Eckmiller (ed.), *Neural Computers*. Springer-Verlag.
